# Using Vocabulary Knowledge in Bayesian Multinomial Estimation

Thomas L. Griffiths & Joshua B. Tenenbaum
Department of Psychology
Stanford University, Stanford, CA 94305
{gruffydd,jbt}@psych.stanford.edu

## Abstract

Estimating the parameters of sparse multinomial distributions is an important component of many statistical learning tasks. Recent approaches have used uncertainty over the vocabulary of symbols in a multinomial distribution as a means of accounting for sparsity. We present a Bayesian approach that allows weak prior knowledge, in the form of a small set of approximate candidate vocabularies, to be used to dramatically improve the resulting estimates. We demonstrate these improvements in applications to text compression and estimating distributions over words in newsgroup data.

## 1   Introduction

Sparse multinomial distributions arise in many statistical domains, including natural language processing and graphical models. Consequently, a number of approaches to parameter estimation for sparse multinomial distributions have been suggested [3]. These approaches tend to be domain-independent: they make little use of prior knowledge about a specific domain. In many domains where multinomial distributions are estimated there is often at least weak prior knowledge about the potential structure of distributions, such as a set of hypotheses about restricted vocabularies from which the symbols might be generated. Such knowledge can be solicited from experts or obtained from unlabeled data. We present a method for Bayesian parameter estimation in sparse discrete domains that exploits this weak form of prior knowledge to improve estimates over knowledge-free approaches.

### 1.1   Bayesian parameter estimation for multinomial distributions

Following the presentation in [4], we consider a language $\Sigma$ containing $L$ distinct symbols. A multinomial distribution is specified by a parameter vector $\theta = \langle \theta_1, \ldots, \theta_L \rangle$, where $\theta_i$ is the probability of an observation being symbol $i$. Consequently, we have the constraints that $\sum_{i=1}^{L} \theta_i = 1$ and $\theta_i \geq 0, i = 1, \ldots, L$. The task of multinomial estimation is to take a data set $D$ and produce a vector $\theta$ that results in a good approximation to the distribution that produced $D$. In this case, $D$ consists of $N$ independent observations $x^1, \ldots x^N$ drawn from the distribution to be estimated, which can be summarized by the statistics $N_i$ specifying the number of times the $i$th symbol occurs in the data. $D$ also determines the set $\Sigma^0$

of symbols that occur in the data.

Stated in this way, multinomial estimation involves predicting the next observation based on the data. Specifically, we wish to calculate $P(x^{N+1}|D)$. The Bayesian estimate for this probability is given by

$$P_L(x^{N+1}|D) = \int P(x^{N+1}|\theta)P(\theta|D)d\theta$$

where $P(x^{N+1}|\theta)$ follows from the multinomial distribution corresponding to $\theta$. The posterior probability $P(\theta|D)$ can be obtained via Bayes rule

$$P(\theta|D) \propto P(D|\theta)P(\theta) = P(\theta)\prod_{i=1}^{L}\theta^{N_i}$$

where $P(\theta)$ is the prior probability of a given $\theta$.

Laplace used this method with a uniform prior over $\theta$ to give the famous "law of succession" [6]. A more general approach is to assume a Dirichlet prior over $\theta$, which is conjugate to the multinomial distribution and gives

$$P(X^{N+1} = i|D) = \frac{N_i + \alpha_i}{N + \sum_{j=1}^{L}\alpha_j} \tag{1}$$

where the $\alpha_i$ are the hyperparameters of the Dirichlet distribution. Different estimates are obtained for different choices of the $\alpha_i$, with most approaches making the simplifying assumption that $\alpha_i = \alpha$ for all $i$. Laplace's law results from $\alpha = 1$. The case with $\alpha = 0.5$ is the Jeffreys-Perks law or Expected Likelihood Estimation [2] [5] [9], while using arbitrary $\alpha$ is Lidstone's law [7].

## 1.2 Estimating sparse multinomial distributions

Several authors have extended the Bayesian approach to sparse multinomial distributions, in which only a restricted vocabulary of symbols are used, by maintaining uncertainty over these vocabularies. In [10], Ristad uses assumptions about the probability of strings based upon different vocabularies to give the estimate

$$P_R(X^{N+1} = i|D) = \begin{cases} (N_i + 1)/(N + L) & \text{if } k^0 = L \\ (N_i + 1)(N + 1 - k^0)/(N^2 + N + 2k^0) & \text{if } k^0 < L \wedge N_i > 0 \\ k^0(k^0 + 1)/(L - k^0)(N^2 + N + 2k^0) & \text{otherwise} \end{cases}$$

where $k^0 = |\Sigma^0|$ is the size of the smallest vocabulary consistent with the data.

A different approach is taken by Friedman and Singer in [4], who point out that Ristad's method is a special case of their framework. Friedman and Singer consider the vocabulary $V \subseteq \Sigma$ to be a random variable, allowing them to write

$$P(X^{N+1} = i|D) = \sum_{V} P(X^{N+1} = i|V, D)P(V|D) \tag{2}$$

where $P(X^{N+1} = i|V, D)$ results from a Dirichlet prior over the symbols in $V$,

$$P(X^{N+1} = i|V, D) = \begin{cases} \frac{N_i + \alpha}{N + |V|\alpha} & \text{if } i \in V \\ 0 & \text{otherwise} \end{cases} \tag{3}$$

and by Bayes' rule and the properties of Dirichlet priors

$$\begin{aligned} P(V|D) &\propto P(D|V)P(V) \\ &= \begin{cases} \frac{\Gamma(|V|\alpha)}{\Gamma(N + |V|\alpha)}\prod_{i\in\Sigma^0}\frac{\Gamma(N_i+\alpha)}{\Gamma(\alpha)}P(V) & \Sigma^0 \subseteq V \\ 0 & \text{otherwise} \end{cases} \end{aligned} \tag{4}$$

Friedman and Singer assume a hierarchical prior over $V$, such that all vocabularies of cardinality $k$ are given equal probability, namely $P(S = k)/\binom{L}{k}$, where $P(S = k)$ is the probability that the size of the vocabulary $(|V|)$ is $k$. It follows that if $i \in \Sigma^0$, $P(X^{N+1} = i|D) = \sum_k \frac{N_i + \alpha}{N + k\alpha} P(S = k|D)$. If $i \notin \Sigma^0$, it is necessary to estimate the proportion of $V$ that contain $i$ for a given $k$. The simplified result is

$$P_F(X^{N+1} = i|D) = \begin{cases} \frac{N_i + \alpha}{N + k^0 \alpha} C(D, L) & \text{if } i \in \Sigma^0 \\ \frac{1}{L - k^0}(1 - C(D, L)) & \text{otherwise} \end{cases} \qquad (5)$$

where

$$C(D, L) = \frac{\sum_{k=k^0}^{L} \frac{N + k^0 \alpha}{N + k\alpha} m_k}{\sum_{k'=k^0}^{L} m_{k'}}$$

with $m_k = P(S = k) \frac{k!}{(k - k^0)!} \cdot \frac{\Gamma(k\alpha)}{\Gamma(N + k\alpha)}$.

## 2    Making use of weak prior knowledge

Friedman and Singer assume a prior that gives equal probability to all vocabularies of a given cardinality. However, many real-world tasks provide limited knowledge about the structure of distributions that we can build into our methods for parameter estimation. In the context of sparse multinomial estimation, one instance of such knowledge the importance of specific vocabularies. For example, in predicting the next character in a file, our predictions could be facilitated by considering the fact that most files either use a vocabulary consisting of ASCII printing characters (such as text files), or all possible characters (such as object files). This kind of structural knowledge about a domain is typically easier to solicit from experts than accurate distributional information, and forms a valuable informational resource.

If we have this kind of prior knowledge, we can restrict our attention to a subset of the $2^L$ possible vocabularies. In particular, we can specify a set of vocabularies $\mathcal{V}$ which we consider as hypotheses for the vocabulary used in producing $D$, where the elements of $\mathcal{V}$ are specified by our knowledge of the domain. This stands as a compromise between Friedman and Singer's approach, in which $\mathcal{V}$ consists of all vocabularies, and traditional Bayesian parameter estimation as represented by Equation 1, in which $\mathcal{V}$ consists of only the vocabulary containing all words. To do this, we explicitly evaluate the sum given in Equation 2, where the sum over $V$ includes all $V \in \mathcal{V}$. This sum remains tractable when $\mathcal{V}$ is a small subset of the possible vocabularies, and the efficiency is aided by the fact that $P(D|V)$ shares common terms across all $V$ which can cancel in normalization.

The intuition behind this approach is that it attempts to classify the target distribution as using one of a known set of vocabularies, where the vocabularies are obtained either from experts or from unlabeled data. Applying standard Bayesian multinomial estimation within this vocabulary gives enough flexibility for the method to capture a range of distributions, while making use of our weak prior knowledge.

### 2.1    An illustration: Text compression

Text compression is an effective test of methods for multinomial estimation. Adaptive coding can be performed by specifying a method for calculating a distribution over the probability of the next byte in a file based upon the preceding bytes [1]. The extent to which the file is compressed depends upon the quality of these predictions. To illustrate the utility of including prior knowledge, we follow Ristad in using the Calgary text compression corpus [1]. This corpus consists of 19 files of

Table 1: Text compression lengths (in bytes) on the Calgary corpus

| file | size | $k^0$ | $NH(N_i/N)$ | $P_V$ | $P_F$ | $P_R$ | $P_L$ | $P_J$ |
|------|------|------|------|------|------|------|------|------|
| bib | 111261 | 81 | 72330 | **78** | 89 | 92 | 269 | 174 |
| book1 | 768771 | 82 | 435043 | 219 | **105** | 116 | 352 | 219 |
| book2 | 610856 | 96 | 365952 | **94** | 115 | 124 | 329 | 212 |
| geo | 102400 | 256 | 72274 | **161** | 162 | 165 | 165 | **161** |
| news | 377109 | 98 | 244633 | **89** | 113 | 116 | 304 | 201 |
| obj1 | 21504 | 256 | 15989 | **126** | 127 | 129 | 129 | **126** |
| obj2 | 246814 | 256 | 193144 | **182** | 184 | 190 | 189 | **182** |
| paper1 | 53161 | 95 | 33113 | **71** | 94 | 100 | 236 | 156 |
| paper2 | 82199 | 91 | 47280 | **75** | 94 | 105 | 259 | 167 |
| paper3 | 46526 | 84 | 27132 | **70** | 85 | 92 | 238 | 154 |
| paper4 | 13286 | 80 | 7806 | **58** | 72 | 79 | 190 | 126 |
| paper5 | 11954 | 91 | 7376 | **57** | 79 | 83 | 181 | 122 |
| paper6 | 38105 | 93 | 23861 | **68** | 90 | 95 | 223 | 149 |
| pic | 513216 | 159 | 77636 | 205 | **162** | 216 | 323 | 205 |
| progc | 39611 | 92 | 25743 | **68** | 89 | 91 | 222 | 150 |
| progl | 71646 | 87 | 42720 | **74** | 91 | 97 | 253 | 164 |
| progp | 49379 | 89 | 30052 | **71** | 89 | 94 | 236 | 155 |
| trans | 93695 | 99 | 64800 | 169 | **101** | 105 | 252 | 169 |

several different types, each using some subset of 256 possible characters ($L = 256$).
The files include BibTEXsource (bib), formatted English text (book*, paper*), ge-
ological data (geo), newsgroup articles (news), object files (obj*), a bit-mapped
picture (pic), programs in three different languages (prog*) and a terminal tran-
script (trans). The task was to estimate the distribution from which characters in
the file were drawn based upon the first $N$ characters and thus predict the $N + 1$st
character. Performance was measured in terms of the length of the resulting file,
where the contribution of the $N + 1$st character to the length is $\log_2 P(x^{N+1}|D)$.
The results are expressed as the number of bytes required to encode the file relative
to the empirical entropy $NH(N_i/N)$ as assessed by Ristad [10].

Results are shown in Table 1. $P_V$ is the restricted vocabulary model outlined above,
with $\mathcal{V}$ consisting of just two hypotheses: one corresponding to binary files, contain-
ing all 256 characters, and one consisting of a 107 character vocabulary representing
formatted English. The latter vocabulary was estimated from 5MB of English text,
C code, BibTEXsource, and newsgroup data from outside the Calgary corpus. $P_F$
is Friedman and Singer's method. For both of these approaches, $\alpha$ was set to 0.5,
to allow direct comparison to the Jeffreys-Perks law, $P_J$. $P_R$ and $P_L$ are Ristad's
and Laplace's laws respectively. $P_V$ outperformed the other methods on all files
based upon English text, bar book1, and all files using all 256 symbols[1]. The high
performance followed from rapid classification of these files as using the appropriate
vocabulary in $\mathcal{V}$. When the vocabulary included all symbols $P_V$ performed as $P_J$,
which gave the best predictions for these files.

those presented here. We have not included these techniques for comparison because our
interest is in using text compression as a means of assessing estimation procedures, rather
than as an end in itself. We thus consider only methods for multinomial estimation as our
comparison group.

## 2.2 Maintaining uncertainty in vocabularies

The results for `book1` illustrate a weakness of the approach outlined above. The file length for $P_V$ is higher than those for $P_F$ and $P_R$, despite the fact that the file uses a text-based vocabulary. This file contains two characters that were not encountered in the data used to construct $\mathcal{V}$. These characters caused $P_V$ to default to the unrestricted vocabulary of all 256 characters. From that point $P_V$ corresponded to $P_J$, which gave poor results on this file.

This behavior results from the assumption that the candidate vocabularies in $\mathcal{V}$ are completely accurate. Since in many cases the knowledge that informs the vocabularies in $\mathcal{V}$ may be imperfect, it is desirable to allow for uncertainty in vocabularies. This uncertainty will be reflected in the fact that symbols outside $V$ are expected to occur with a vocabulary-specific probability $\epsilon_V$,

$$P(X^{N+1} = i|V, D) = \begin{cases} (1 - (L - |V|)\epsilon_V)\frac{N_i + \alpha}{N_V + |V|\alpha} & \text{if } i \in V \\ \epsilon_V & \text{otherwise} \end{cases}$$

where $N_V = \sum_{i \in V} N_i$. It follows that

$$P(D|V) = (1 - (L - |V|)\epsilon_V)^{N_V} \epsilon_V^{N - N_V} \frac{\Gamma(|V|\alpha)}{\Gamma(N_V + |V|\alpha)} \prod_{i \in \Sigma_0 \cap V} \frac{\Gamma(N_i + \alpha)}{\Gamma(\alpha)}$$

which replaces Equations 3-4 in specifying $P_V$.

When $\mathcal{V}$ is determined by the judgments of domain experts, $\epsilon_V$ is the probability that an unmentioned word actually belongs to a particular vocabulary. While it may not be the most efficient use of such data, the $V \in \mathcal{V}$ can also be estimated from some form of unlabeled data. In this case, Friedman and Singer's approach provides a means of setting $\epsilon_V$. Friedman and Singer explicitly calculate the probability that an unseen word is in $V$ based upon a dataset: from the second condition of Equation 5, we find that we should set $\epsilon_V = \frac{1}{L - |V|}(1 - C(D, L))$. We use this method below.

## 3  Bayesian parameter estimation in natural language

Statistical natural language processing often uses sparse multinomial distributions over large vocabularies of words. In different contexts, different vocabularies will be used. By specifying a basis set of vocabularies, we can perform parameter estimation by classifying distributions according to their vocabulary. This idea was examined using data from 20 different Usenet newsgroups. This dataset is commonly used in testing text classification algorithms (eg. [8]). Ten newsgroups were used to estimate a set of vocabularies $\mathcal{V}$ with corresponding $\epsilon_V$. These vocabularies were used in estimating multinomial distributions on these newsgroups and ten others.

The dataset was `20news-18827`, which consists of the `20newsgroups` data with headers and duplicates removed, and was preprocessed to remove all punctuation, capitalization, and distinct numbers. The articles in each of the 20 newsgroups were then divided into three sets. The first 500 articles from ten newsgroups were used to estimate the candidate vocabularies $\mathcal{V}$ and uncertainty parameters $\epsilon_V$. Articles 501-700 for all 20 newsgroups were used as training data for multinomial estimation. Articles 701-900 for all 20 newgroups were used as testing data. Following [8], a dictionary was built up by running over the 13,000 articles resulting from this division, and all words that occurred only once were mapped to an "unknown" word. The resulting dictionary contained $L = 54309$ words.

As before, the restricted vocabulary method ($P_V$), Friedman and Singer's method ($P_F$), and Ristad's ($P_R$), Laplace's ($P_L$) and the Jeffreys-Perks ($P_J$) laws were ap-

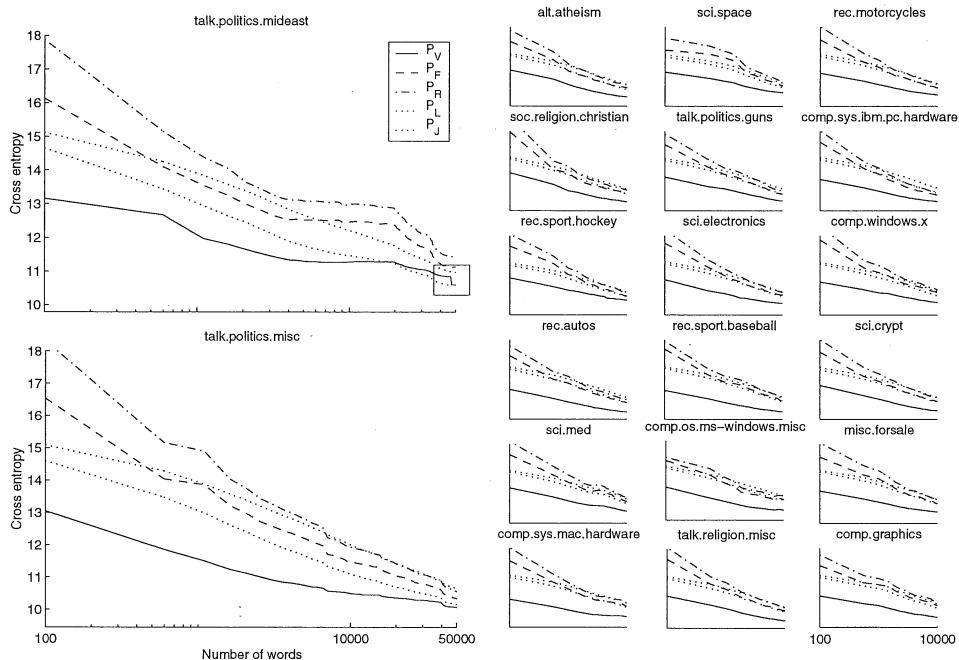

Figure 1: Cross-entropy of predictions on newsgroup data as a function of the logarithm of the number of words. The abscissa is at the empirical entropy of the test distribution. The top ten panels (talk.politics.mideast and those to its right) are for the newsgroups with unknown vocabularies. The bottom ten are for those that contributed vocabularies to $\mathcal{V}$, trained and tested on novel data. $P_L$ and $P_J$ are both indicated with dotted lines, but $P_J$ always performs better than $P_L$. The box on talk.politics.mideast indicates the point at which $P_V$ defaults to the full vocabulary, as the number of unseen words makes this vocabulary more likely. At this point, the line for $P_V$ joins the line for $P_J$, since both methods give the same estimates of the distribution.

plied to the task. Both $P_V$ and $P_F$ used $\alpha = 0.5$ to facilitate comparison with $P_J$. $\mathcal{V}$ featured one vocabulary that contained all words in the dictionary, and ten vocabularies each corresponding to the words used in the first 500 articles of one of the newsgroups designated for this purpose. $\epsilon_V$ was estimated as outlined above. Testing for each newsgroup consisted of taking words from the 200 articles assigned for training purposes, estimating a distribution using each method, and then computing the cross-entropy between that distribution and an empirical estimate of the true distribution. The cross-entropy is $H(Q; \hat{P}) = \sum_i Q_i \log_2 \hat{P}_i$, where $Q$ is the true distribution and $\hat{P}$ is the distribution produced by the estimation method. $Q$ was given by the maximum likelihood estimate formed from the word frequencies in all 200 articles assigned for testing purposes. The testing procedure was conducted with just 100 words, and then in increments of 450 up to a total of 10000 words. Long-run performance was examined on talk.politics.mideast and talk.politics.misc, each trained with 50000 words.

The results are shown in Figure 1. As expected, $P_V$ consistently outperformed the other methods on the newsgroups that contributed to $\mathcal{V}$. However, performance on novel newsgroups was also greatly improved. As can be seen in Figure 2, the novel newsgroups were classified to appropriate vocabularies – for example

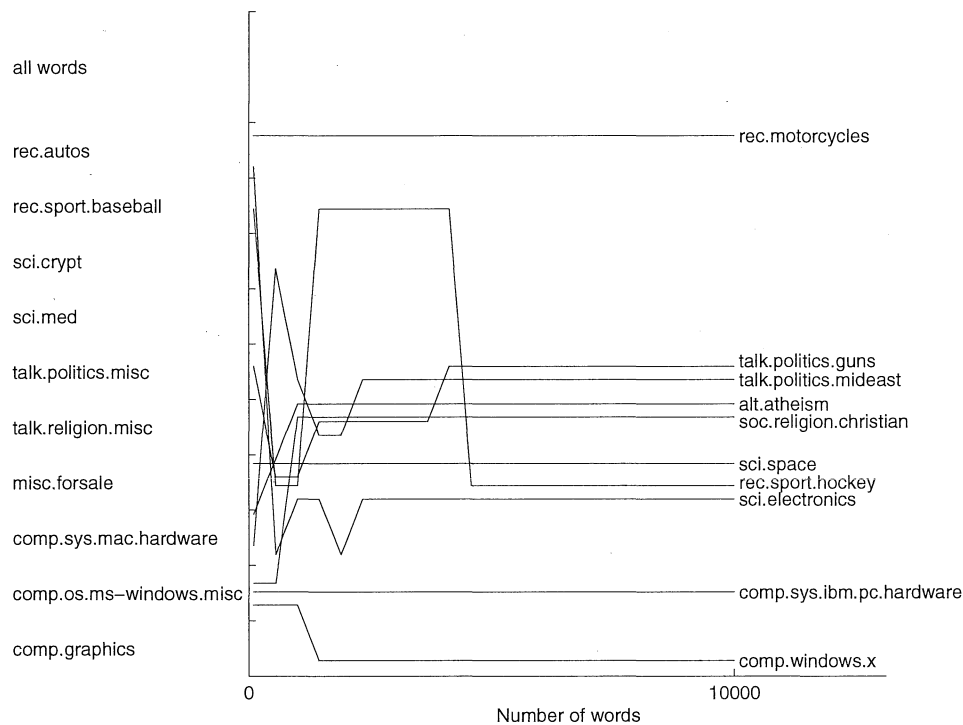

Figure 2: Classification of newsgroup vocabularies. The lines illustrate the vocabulary which had maximum posterior probability for each of the ten test newsgroups after exposure to differing numbers of words. The vocabularies in $\mathcal{V}$ are listed along the left hand side of the axis, and the lines are identified with newsgroups by the labels on the right hand side. Lines are offset to facilitate identification.

talk.religion.misc had the highest posterior probability for alt.atheism and soc.religion.christian, while rec.autos had highest posterior probability for rec.motorcycles. The proportion of word types occurring in the test data but not the vocabulary to which the novel newsgroups were classified ranged between 30.5% and 66.2%, with a mean of 42.2%. This illustrates that even approximate knowledge can facilitate predictions: the basis set of vocabularies allowed the high frequency words in the data to be modelled effectively, without excess mass being attributed to the low frequency novel word tokens.

Long-run performance on talk.politics.mideast illustrates the same defaulting behavior that was shown for text classification: when the data become more probable under the vocabulary containing all words than under a restricted vocabulary the method defaults to the Jeffreys-Perks law. This guarantees that the method will tend to perform no worse than $P_J$ when unseen words are encountered in sufficient proportions. This is desirable, since $P_J$ gives good estimates once $N$ becomes large.

## 4    Discussion

Bayesian approaches to parameter estimation for sparse multinomial distributions have employed the notion of a restricted vocabulary from which symbols are produced. In many domains where such distributions are estimated, there is often at

least limited knowledge about the structure of these vocabularies. By considering just the vocabularies suggested by such knowledge, together with some uncertainty concerning those vocabularies, we can achieve very good estimates of distributions in these domains. We have presented a Bayesian approach that employs limited prior knowledge, and shown that it outperforms a range of approaches to multinomial estimation for both text compression and a task involving natural language.

While our applications in this paper estimated approximate vocabularies from data, the real promise of this approach lies with domain knowledge solicited from experts. Experts are typically better at providing qualitative structural information than quantitative distributional information, and our approach provides a way of using this information in parameter estimation. For example, in the context of parameter estimation for graphical models to be used in medical diagnosis, distinguishing classes of symptoms might be informative in determining the parameters governing their relationship to diseases. This form of knowledge is naturally translated into a set of vocabularies to be considered for each such distribution. More complex applications to natural language may also be possible, such as using syntactic information in estimating probabilities for $n$-gram models. The approach we have presented in this paper provides a simple way to allow this kind of limited domain knowledge to be useful in Bayesian parameter estimation.

## Footnotes

[1]A number of excellent techniques for text compression exist that outperform all of

# References

[1] T. C. Bell, J. G. Cleary, and I. H. Witten. *Text compression*. Prentice Hall, 1990.

[2] G. E. P. Box and G. C. Tiao. *Bayesian Inference in Statistical Analysis*. Addison-Wesley, 1973.

[3] S. F. Chen and J. Goodman. An empirical study of smoothing techniques for language modeling. Technical Report TR-10-98, Center for Research in Computing Technology, Harvard University, 1998.

[4] N. Friedman and Y. Singer. Efficient Bayesian parameter estimation in large discrete domains. In *Neural Information Processing Systems*, 1998.

[5] H. Jeffreys. An invariant form for the prior probability in estimation problems. *Proceedings of the Royal Society A*, 186:453–461, 1946.

[6] P.-S. Laplace. *Philosophical Essay on Probabilities*. Springer-Verlag, 1995. Originally published 1825.

[7] G. Lidstone. Note on the general case of the Bayes-Laplace formula for inductive or a posteriori probabilities. *Transactions of the Faculty of Actuaries*, 8:182–192, 1920.

[8] K. Nigam, A. K. Mccallum, S. Thrun, and T. Mitchell. Text classification from labeled and unlabeled documents using EM. *Machine Learning*, 39:103–134, 2000.

[9] W. Perks. Some observations on inverse probability, including a new indifference rule. *Journal of the Institute of Actuaries*, 73:285–312, 1947.

[10] E. S. Ristad. A natural law of succession. Technical Report CS-TR-895-95, Department of Computer Science, Princeton University, 1995.
